# The Noisy-Logical Distribution and its Application to Causal Inference

**Alan Yuille**
Department of Statistics
University of California at Los Angeles
Los Angeles, CA 90095
yuille@stat.ucla.edu

**Hongjing Lu**
Department of Psychology
University of California at Los Angeles
Los Angeles, CA 90095
hongjing@ucla.edu

## Abstract

We describe a novel noisy-logical distribution for representing the distribution of a binary output variable conditioned on multiple binary input variables. The distribution is represented in terms of noisy-or's and noisy-and-not's of *causal features* which are conjunctions of the binary inputs. The standard noisy-or and noisy-and-not models, used in causal reasoning and artificial intelligence, are special cases of the noisy-logical distribution. We prove that the noisy-logical distribution is complete in the sense that it can represent all conditional distributions provided a sufficient number of causal factors are used. We illustrate the noisy-logical distribution by showing that it can account for new experimental findings on how humans perform causal reasoning in complex contexts. We speculate on the use of the noisy-logical distribution for causal reasoning and artificial intelligence.

## 1 Introduction

The noisy-or and noisy-and-not conditional probability distributions are frequently studied in cognitive science for modeling causal reasoning [1], [2],[3] and are also used as probabilistic models for artificial intelligence [4]. It has been shown, for example, that human judgments of the power of causal cues in experiments involving two cues [1] can be interpreted in terms of maximum likelihood estimation and model selection using these types of models [3].

But the noisy-or and noisy-and-not distributions are limited in the sense that they can only represent a restricted set of all possible conditional distributions. This restriction is sometimes an advantage because there may not be sufficient data to determine the full conditional distribution. Nevertheless it would be better to have a representation that can expand to represent the full conditional distribution, if sufficient data is available, but can be reduced to simpler forms (e.g. standard noisy-or) if there is only limited data.

This motivates us to define the noisy-logical distribution. This is defined in terms of noisy-or's and noisy-and-not's of *causal features* which are conjunctions of the basic input variables (inspired by the use of conjunctive features in [2] and the extensions in [5]). By restricting the choice of causal features we can obtain the standard noisy-or and noisy-and-not models. We prove that the noisy-logical distribution is complete in the sense that it can represent any conditional distribution provided we use all the causal features. Overall, it gives a distribution whose complexity can be adjusted by restricting the number of causal features.

To illustrate the noisy-logical distribution we apply it to modeling some recent human experiments on causal reasoning in complex environments [6]. We show that noisy-logical distributions involving causal factors are able to account for human performance. By contrast, an alternative linear model gives predictions which are the opposite of the observed trends in human causal judgments. Section (2) presents the noisy-logical distribution for the case with two input causes (the case commonly studied in causal reasoning). In section (3) we specify the full noisy-logical distribution and

we prove its completeness in section (4). Section (5) illustrates the noisy-logical distribution by showing that it accounts for recent experimental findings in causal reasoning.

## 2 The Case with $N = 2$ causes

In this section we study the simple case when the binary output effect $E$ depends only on two binary-valued causes $C_1, C_2$. This covers most of the work reported in the cognitive science literature [1],[3]. In this case, the probability distribution is specified by the four numbers $P(E = 1|C_1, C_2)$, for $C_1 \in \{0, 1\}$, $C_2 \in \{0, 1\}$.

To define the noisy-logical distribution over two variables $P(E = 1|C_1, C_2)$, we introduce three concepts. Firstly, we define four binary-valued causal features $\Psi_0(.), \Psi_1(.), \Psi_2(.), \Psi_3(.)$ which are functions of the input state $\vec{C} = (C_1, C_2)$. They are defined by $\Psi_0(\vec{C}) = 1, \Psi_1(\vec{C}) = C_1, \Psi_2(\vec{C}) = C_2, \Psi_3(\vec{C}) = C_1 \wedge C_2$, where $\wedge$ denotes logical-and operation(i.e. $C_1 \wedge C_2 = 1$ if $C_1 = C_2 = 1$ and $C_1 \wedge C_2 = 0$ otherwise). $\Psi_3(\vec{C})$ is the conjunction of $C_1$ and $C_2$. Secondly, we introduce binary-valued hidden states $E_0, E_1, E_2, E_3$ which are caused by the corresponding features $\Psi_0, \Psi_1, \Psi_2, \Psi_3$. We define $P(E_i = 1|\Psi_i; \omega_i) = \omega_i \Psi_i$ with $\omega_i \in [0, 1]$, for $i = 1, ..., 4$ with $\vec{\omega} = (\omega_1, \omega_2, \omega_3, \omega_4)$. Thirdly, we define the output effect $E$ to be a logical combination of the states $E_0, E_1, E_2, E_3$ which we write in form $\delta_{E, f(E_0, E_1, E_2, E_3)}$, where $f(., ., ., .)$ is a logic function which is formed by a combination of three logic operations $AND, OR, NOT$. This induces the *noisy-logical distribution* $P_{nl}(E|\vec{C}; \vec{\omega}) = \sum_{E_0, ..., E_3} \delta_{E, f(E_0, E_1, E_2, E_3)} \prod_{i=0}^{3} P(E_i|\Psi_i(\vec{C}); \omega_i)$.

The noisy-logical distribution is characterized by the parameters $\omega_0, ..., \omega_3$ and the choice of the logic function $f(., ., ., .)$. We can represent the distribution by a circuit diagram where the output $E$ is a logical function of the hidden states $E_0, ..., E_3$ and each state is caused probabilistically by the corresponding causal features $\Psi_0, ..., \Psi_3$, as shown in Figure (1).

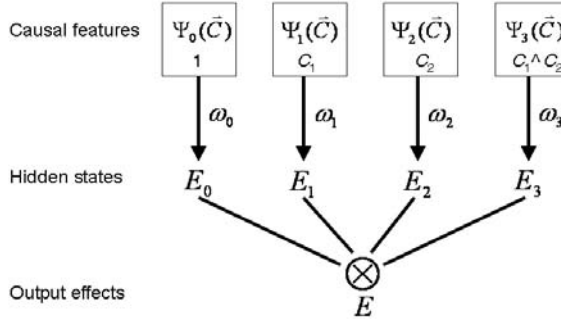

Figure 1: Circuit diagram in the case with $N = 2$ causes.

The noisy-logical distribution includes the commonly known distributions, noisy-or and noisy-and-not, as special cases. To obtain the noisy-or, we set $E = E_1 \vee E_2$ (i.e. $E_1 \vee E_2 = 0$ if $E_1 = E_2 = 0$ and $E_1 \vee E_2 = 1$ otherwise). A simple calculation shows that the noisy-logical distribution reduces to the noisy-or $P_{nor}(E|C_1, C_2; \omega_1, \omega_2)$ [4], [1]:

$$
\begin{aligned}
P_{nl}(E = 1|C_1, C_2; \omega_1, \omega_2) &= \sum_{E_1, E_2} \delta_{1, E_1 \vee E_2} P(E_1|\Psi_1(\vec{C}); \omega_1) P(E_2|\Psi_2(\vec{C}); \omega_2) \\
&= \omega_1 C_1 (1 - \omega_2 C_2) + (1 - \omega_1 C_1) \omega_2 C_2 + \omega_1 \omega_2 C_1 C_2 \\
&= \omega_1 C_1 + \omega_2 C_2 - \omega_1 \omega_2 C_1 C_2 = P_{nor}(E = 1|C_1, C_2; \omega_1, \omega_2) \quad (1)
\end{aligned}
$$

To obtain the noisy-and-not, we set $E = E_1 \wedge \neg E_2$ (i.e. $E_1 \wedge \neg E_2 = 1$ if $E_1 = 1, E_2 = 0$ and $E_1 \wedge \neg E_2 = 0$ otherwise). The noisy-logical distribution reduces to the noisy-and-not $P_{n-and-not}(E|C_1, C_2; \omega_1, \omega_2)$ [4],[?]:

$$
\begin{aligned}
P_{nl}(E = 1|C_1, C_2; \omega_1, \omega_2) &= \sum_{E_1, E_2} \delta_{1, E_1 \wedge \neg E_2} P(E_1|\Psi_1(\vec{C}); \omega_1) P(E_2|\Psi_2(\vec{C}); \omega_2) \\
&= \omega_1 C_1 \{1 - \omega_2 C_2\} = P_{n-and-not}(E = 1|C_1, C_2; \omega_1, \omega_2) \quad (2)
\end{aligned}
$$

We claim that noisy-logical distributions of this form can represent any conditional distribution $P(E|\vec{C})$. The logical function $f(E_0, E_1, E_2, E_3)$ will be expressed as a combination of logic operations AND-NOT, OR. The parameters of the distribution are given by $\omega_0, \omega_1, \omega_2, \omega_3$.

The proof of this claim will be given for the general case in the next section. To get some insight, we consider the special case where we only know the values $P(E|C_1 = 1, C_2 = 0)$ and $P(E|C_1 = 1, C_2 = 1)$. This situation is studied in cognitive science where $C_1$ is considered to be a background cause which always takes value 1, see [1] [3]. In this case, the only causal features are considered, $\Psi_1(\vec{C}) = C_1$ and $\Psi_2(\vec{C}) = C_2$.

**Result**. *The noisy-or and the noisy-and-not models, given by equations (1,2) are sufficient to fit any values of $P(E = 1|1, 0)$ and $P(E = 1|1, 1)$. (In this section we use $P(E = 1|1, 0)$ to denote $P(E = 1|C_1 = 1, C_2 = 0)$ and use $P(E = 1|1, 1)$ to denote $P(E = 1|C_1 = 1, C_2 = 1)$.) The noisy-or and noisy-and-not fit the cases when $P(E = 1|1, 1) \geq P(E = 1|1, 0)$ and $P(E = 1|1, 1) \leq P(E = 1|1, 0)$ respectively. In Cheng's terminology [1] $C_2$ is respectively a* generative *or* preventative *cause).*

Proof. *We can fit both the noisy-or and noisy-and-not models to $P(E|1, 0)$ by setting $\omega_1 = P(E = 1|1, 0)$, so it remains to fit the models to $P(E|1, 1)$. There are three cases to consider: (i) $P(E = 1|1, 1) > P(E = 1|1, 0)$, (ii) $P(E = 1|1, 1) < P(E = 1|1, 0)$, and (iii) $P(E = 1|1, 1) = P(E = 1|1, 0)$. It follows directly from equations (1,2) that $P_{nor}(E = 1|1, 1) \geq P_{nor}(E = 1|1, 0)$ and $P_{n-and-not}(E = 1|1, 1) \leq P_{n-and-not}(E = 1|1, 0)$ with equality only if $P(E = 1|1, 1) = P(E = 1|1, 0)$. Hence we must fit a noisy-or and a noisy-and-not model to cases (i) and (ii) respectively. For case (i), this requires solving $P(E = 1|1, 1) = \omega_1 + \omega_2 - \omega_1\omega_2$ to obtain $\omega_2 = \{P(E = 1|1, 1) - P(E = 1|1, 0)\}/\{1 - P(E = 1|1, 0)\}$ (note that the condition $P(E = 1|1, 1) > P(E = 1|1, 0)$ ensures that $\omega_2 \in [0, 1]$). For case (ii), we must solve $P(E = 1|1, 1) = \omega_1 - \omega_1\omega_2$ which gives $\omega_2 = \{P(E = 1|1, 0) - P(E = 1|1, 1)\}/P(E = 1|1, 0)$ (the condition $P(E = 1|1, 1) < P(E = 1|1, 0)$ ensures that $\omega_2 \in [0, 1]$). For case (iii), we can fit either model by setting $\omega_2 = 0$.*

## 3  The Noisy-Logical Distribution for $N$ causes

We next consider representing probability distributions of form $P(E|\vec{C})$, where $E \in \{0, 1\}$ and $\vec{C} = (C_1, ..., C_N)$ where $C_i \in \{0, 1\}$, $\forall i = 1, .., N$. These distributions can be characterized by the values of $P(E = 1|\vec{C})$ for all possible $2^N$ values of $\vec{C}$.

We define the set of $2^N$ binary-valued causal features $\{\Psi_i(\vec{C}) : i = 0, ..., 2^N - 1\}$. These features are ordered so that $\Psi_0(\vec{C}) = 1$, $\Psi_i(\vec{C}) = C_i : i = 1, .., N$, $\Psi_{N+1}(\vec{C}) = C_1 \wedge C_2$ is the conjunction of $C_1$ and $C_2$, and so on. The feature $\Psi(\vec{C}) = C_a \wedge C_b \wedge ... \wedge C_g$ will take value 1 if $C_a = C_b = ... = C_g = 1$ and value 0 otherwise.

We define binary variables $\{E_i : i = 0, ..., 2^N - 1\}$ which are related to the causal features $\{\Psi_i : i = 0, ..., 2^N - 1\}$ by distributions $P(E_i = 1|\Psi_i; \omega_i) = \omega_i\Psi_i$, specified by parameters $\{\omega_i : i = 0, ..., 2^N - 1\}$.

Then we define the output variable $E$ to be a logical (i.e. deterministic) function of the $\{E_i : i = 0, ..., 2^N - 1\}$. This can be thought of as a circuit diagram. In particular, we define $E = f(E_0, ..., E_{2^N-1}) = (((((E_1 \otimes E_2) \otimes E_3) \otimes E_4....)$ where $E_1 \otimes E_2$ can be $E_1 \vee E_2$ or $E_1 \wedge \neg E_2$ (where $\neg E$ means logical negation). This gives the general *noisy-logical distribution*, as shown in Figure (2).

$$P(E = 1|\vec{C}; \vec{\omega}) = \sum_{\vec{E}} \delta_{E, f(E_0, ..., E_{2^N-1})} \prod_{i=0}^{2^N-1} P(E_i = 1|\Psi_i; \omega_i). \qquad (3)$$

## 4  The Completeness Result

This section proves that the noisy-logical distribution is capable of representing any conditional distribution. This is the main theoretical result of this paper.

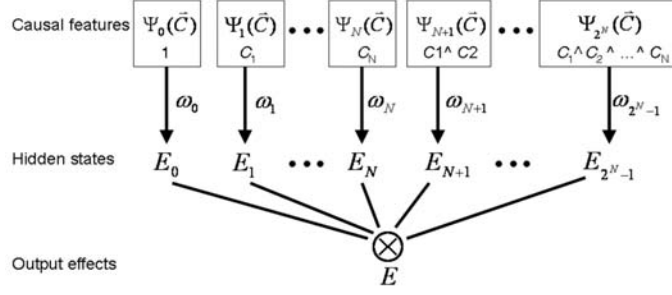

Figure 2: Circuit diagram in the case with N causes. All conditional distributions can be represented in this form if we use all possible $2^N$ causal features $\Psi$, choose the correct parameters $\omega$, and select the correct logical combinations $\otimes$.

**Result** *We can represent any conditional distribution $P(E|\vec{C})$ defined on binary variables in terms of a noisy logical distribution given by equation (3).*

Proof. *The proof is constructive. We show that any distribution $P(E|\vec{C})$ can be expressed as a noisy-logical distribution.*

*We order the states $\vec{C}_0, ..., \vec{C}_{2^N-1}$. This ordering must obey $\Psi_i(\vec{C}_i) = 1$ and $\Psi_i(\vec{C}_j) = 0$, $\forall j < i$. This ordering can be obtained by setting $\vec{C}_0 = (0, ..., 0)$, then selecting the terms with a single conjunction (i.e. only one $C_i$ is non-zero), then those with two conjunctions (i.e. two $C_i$'s are non-zero), then with three conjunctions, and so on.*

*The strategy is to use induction to build a noisy-logical distribution which agrees with $P(E|\vec{C})$ for all values of $\vec{C}$. We loop over the states and incrementally construct the logical function $f(E_0, ..., E_{2^N-1})$ and estimate the parameters $\omega_0, ..., \omega_{2^N-1}$. It is convenient to recursively define a variable $E^{i+1} = E^i \otimes E_i$, so that $f(E_0, ..., E_{2^N-1}) = E^{2^N-1}$.*

*We start the induction using feature $\Psi_0(\vec{C}) = 1$. Set $E^0 = E_0$ and $\omega_0 = P(E|0, ..., 0)$. Then $P(E^0|\vec{C}_0; \omega_0) = P(E|\vec{C}_0)$, so the noisy-logical distribution fits the data for input $\vec{C}_0$.*

*Now proceed by induction to determine $E^{M+1}$ and $\omega_{M+1}$, assuming that we have determined $E^M$ and $\omega_0, ..., \omega_M$ such that $P(E^M = 1|\vec{C}_i; \omega_0, ..., \omega_M) = P(E = 1|\vec{C}_i)$, for $i = 0, ..., M$. There are three cases to consider which are analogous to the cases considered in the section with two causes.*

*Case 1. If $P(E = 1|\vec{C}_{M+1}) > P(E^M = 1|\vec{C}_{M+1}; \omega_0, ..., \omega_M)$ we need $\Psi_{M+1}(\vec{C})$ to be a generative feature. Set $E^{M+1} = E^M \vee E_{M+1}$ with $P(E_{M+1} = 1|\Psi_{M+1}; \omega_{M+1}) = \omega_{M+1}\Psi_{M+1}$. Then we obtain:*

$$P(E^{M+1} = 1|\vec{C}_{M+1}; \omega_0, ., \omega_{M+1}) = P(E^M = 1|\vec{C}_{M+1}; \omega_0, ., \omega_M) + P(E_{M+1}|\Psi_{M+1}(\vec{C}); \omega_{M+1})$$

$$-P(E^M = 1|\vec{C}_{M+1}; \omega_0, ., \omega_M)P(E_{M+1} = 1|\Psi_{M+1}(\vec{C}); \omega_{M+1}) =$$

$$P(E^M = 1|\vec{C}_{M+1}; \omega_0, ., \omega_M) + \omega_{M+1}\Psi_{M+1}(\vec{C}) - P(E^M = 1|\vec{C}_{M+1}; \omega_0, ., \omega_M)\omega_{M+1}\Psi_{M+1}(\vec{C})$$

*In particular, we see that $P(E^{M+1} = 1|\vec{C}_i; \omega_0, ..., \omega_{M+1}) = P(E^M = 1|\vec{C}_i; \omega_0, ..., \omega_M) = P(E = 1|\vec{C}_i)$ for $i < M + 1$ (using $\Psi_{M+1}(\vec{C}_i) = 0$, $\forall i < M + 1$). To determine the value of $\omega_{M+1}$, we must solve $P(E = 1|\vec{C}_{M+1}) = P(E^M = 1|\vec{C}_{M+1}; \omega_0, ..., \omega_M) + \omega_{M+1} - P(E^M = 1|\vec{C}_{M+1}; \omega_0, ..., \omega_M)\omega_{M+1}$ (using $\Psi_{M+1}(\vec{C}_{M+1}) = 1$). This gives $\omega_{M+1} = \{P(E = 1|\vec{C}_{M+1}) - P(E^M = 1|\vec{C}_{M+1}; \omega_0, ..., \omega_M)\}/\{1 - P(E^M = 1|\vec{C}_{M+1}; \omega_0, ..., \omega_{M+1})\}$ (the conditions ensure that $\omega_{M+1} \in [0, 1]$).*

*Case 2. If $P(E = 1|\vec{C}_{M+1}) < P(E^M = 1|\vec{C}_{M+1}; \omega_0, ..., \omega_M)$ we need $\Psi_{M+1}(\vec{C})$ to be a preventative feature. Set $E^{M+1} = E^M \wedge \neg E_{M+1}$ with $P(E_{M+1} = 1|\Psi_{M+1}; \omega_{M+1}) = \omega_{M+1}\Psi_{M+1}$. Then we obtain:*

$$P(E^{M+1} = 1|\vec{C}_{M+1}; \omega_0, ..., \omega_{M+1}) = P(E^M = 1|\vec{C}_{M+1}; \omega_0, ..., \omega_M)\{1 - \omega_{M+1}\Psi_{M+1}(\vec{C})\}.$$

$$(4)$$

As for the first case, $P(E^{M+1} = 1|\vec{C}_i; \omega_0, ..., \omega_{M+1}) = P(E^M = 1|\vec{C}_i; \omega_0, ..., \omega_M) = P(E = 1|\vec{C}_i)$ for $i < M + 1$ (because $\Psi_{M+1}(\vec{C}_i) = 0, \forall i < M + 1$). To determine the value of $\omega_{M+1}$ we must solve $P(E = 1|\vec{C}_{M+1}) = P(E^M = 1|\vec{C}_{M+1}; \omega_0, ..., \omega_M)\{1 - \omega_{M+1}\}$ (using $\Psi_{M+1}(\vec{C}_{M+1}) = 1$). This gives $\omega_{M+1} = \{P(E^M = 1|\vec{C}_{M+1}; \omega_0, ..., \omega_M) - P(E = 1|\vec{C}_{M+1})\}/P(E^M = 1|\vec{C}_{M+1}; \omega_0, ..., \omega_M)$ (the conditions ensure that $\omega_{M+1} \in [0, 1]$).

Case 3. If $P(E = 1|\vec{C}_{M+1}) = P(E^M = 1|\vec{C}_{M+1}; \omega_0, ..., \omega_M)$, then we do nothing.

# 5 Cognitive Science Human Experiments

We illustrate noisy-logical distributions by applying them to model two recent cognitive science experiments by Liljeholm and Cheng which involve causal reasoning in complex environments [6]. In these experiments, the participants are asked questions about the causal structure of the data. But the participants are not given enough data to determine the full distribution (i.e. not enough to determine the causal structure with certainty). Instead the experimental design forces them to choose between two different causal structures.

We formulate this as a model selection problem [3]. Formally, we specify distributions $P(D|\vec{\omega}, Graph)$ for generating the data $D$ from a causal model specified by $Graph$ and parameterized by $\vec{\omega}$. These distributions will be of simple noisy-logical form. We set the prior distributions $P(\vec{\omega}|Graph)$ on the parameter values to be the uniform distribution. The evidence for the causal model is given by:

$$P(D|Graph) = \int d\vec{\omega} P(D|\vec{\omega}, Graph) P(\vec{\omega}|Graph). \tag{5}$$

We then evaluate the log-likelihood ratio $\log \frac{P(D|Graph1)}{P(D|Graph2)}$ between two causal models $Graph1$ $Graph2$, called the *causal support* [3] and use this to predict the performance of the participants. This gives good fits to the experimental results.

As an alternative theoretical model, we consider the possibility that the participants use the same causal structures, specified by $Graph1$ and $Graph2$, but use a *linear* model to combine cues. Formally, this corresponds to a model $P(E = 1|C_1, ..., C_N) = \omega_1 C_1 + ... + \omega_N C_N$ (with $\omega_i \geq 0, \forall i = 1, ..., N$ and $\omega_1 + ... + \omega_N \leq 1$). This model corresponds [1, 3] to the classic Rescorla-Wagner learning model [8]. It cannot be expressed in simple noisy-logical form. Our simulations show that this model does not account for human participant performance .

We note that previous attempts to model experiments with multiple causes and conjunctions by Novick and Cheng [2] can be interpreted as performing maximum likelihood estimation of the parameters of noisy-logical distributions (their paper helped inspire our work). Those experiments, however, were simpler than those described here and model selection was not used. The extensive literatures on two cases [1, 3] can also be interpreted in terms of noisy-logical models.

## 5.1 Experiment I: Multiple Causes

In Experiment 1 of [6], the cover story involves a set of allergy patients who either did or did not have a headache, and either had or had not received allergy medicines $A$ and $B$. The experimental participants were informed that two independent studies had been conducted in different labs using different patient groups. In the first study, patients were administered medicine $A$, whereas in the second study patients were administered both medicines $A$ and $B$. A simultaneous presentation format [7] was used to display the specific contingency conditions used in both studies to the experimental subjects. The participants were then asked whether medicine $B$ caused the headache.

We represent this experiment as follows using binary-valued variables $E, B_1, B_2, C_1, C_2$. The variable $E$ indicates whether a headache has occurred ($E = 1$) or not ($E = 0$). $B_1 = 1$ and $B_2 = 1$ notate background causes for the two studies (which are always present). $C_1$ and $C_2$ indicate whether medicine $A$ and $B$ are present respectively (e.g. $C_1 = 1$ if $A$ is present, $C_1 = 0$ otherwise). The data $D$ shown to the subjects can be expressed as $D = (D_1, D_2)$ where $D_1$ is the contingency table $P_d(E = 1|B_1 = 1, C_1 = 0, C_2 = 0), P_d(E = 1|B_1 = 1, C_1 = 1, C_2 = 0)$ for the first study

and $D_2$ is the contingency table $P_d(E = 1|B_2 = 1, C_1 = 0, C_2 = 0), P_d(E = 1|B_2 = 1, C_1 = 1, C_2 = 1)$ for the second study.

The experimental design forces the participants to choose between the two causal models shown on the left of figure (3). These causal models differ by whether $C_2$ (i.e. medicine $B$) can have an effect or not. We set $P(D|\vec{\omega}, Graph) = P(D_1|\vec{\omega}_1, Graph)P(D_2|\vec{\omega}_2, Graph)$, where $D_i = \{(E^\mu, \vec{C}_i^\mu)\}$ (for $i = 1, 2$) is the contingency data. We express these distributions in form $P(D_i|\vec{\omega}_i, Graph) = \prod_\mu P_i(E^\mu|\vec{C}_i^\mu, \vec{\omega}_i^\mu, Graph)$. For $Graph1$, $P_1(.)$ and $P_2(.)$ are $P(E|B_1, C_1, \omega_{B1}, \omega_{C1})$ and $P(E|B_2, C_1, \omega_{B2}, \omega_{C1})$. For $Graph2$, $P_1(.)$ and $P_2(.)$ are $P(E|B_1, C_1, \omega_{B1}, \omega_{C1})$ and $P(E|B_2, C_1, C_2, \omega_{B2}, \omega_{C1}, \omega_{C2})$. All these $P(E|.)$ are noisy-or distributions.

For Experiment 1 there are two conditions [6], see table (1). In the first *power-constant condition* [6], the data is consistent with the causal structure for $Graph1$ (i.e. $C_2$ has no effect) using noisy-or distributions. In the second $\Delta P$-*constant condition* [6], the data is consistent with the causal structure for $Graph1$ but with noisy-or replaced by the linear distributions (e.g. $P(E = 1|C_1, ..., C_n) = \omega_1 C_1 + ... + \omega_n C_n$)).

Table 1: Experimental conditions (1) and (2) for Experiment 1

| | | |
|---|---|---|
| (1) | $P_d(E = 1|B_1 = 1, C_1 = 0, C_2 = 0), P_d(E = 1|B_1 = 1, C_1 = 1, C_2 = 0)$ | 16/24, 22/24 |
| | $P_d(E = 1|B_2 = 1, C_1 = 0, C_2 = 0), P_d(E = 1|B_2 = 1, C_1 = 1, C_2 = 1)$ | 0/24,18/24 |
| (2) | $P_d(E = 1|B_1 = 1, C_1 = 0, C_2 = 0), P_d(E = 1|B_1 = 1, C_1 = 1, C_2 = 0)$ | 0/24, 6/24 |
| | $P_d(E = 1|B_2 = 1, C_1 = 0, C_2 = 0), P_d(E = 1|B_2 = 1, C_1 = 1, C_2 = 1)$ | 16/24,22/24 |

## 5.2 Experiment I: Results

We compare Liljeholm and Cheng's experimental results with our theoretical simulations. These comparisons are shown on the right-hand-side of figure (3). The left panel shows the proportion of participants who decide that medicine $B$ causes a headache for the two conditions. The right panel shows the predictions of our model (labeled "noisy-logical") together with predictions of a model that replaces the noisy-logical distributions by a linear model (labeled "linear"). The simulations show that the noisy-logical model correctly predicts that participants (on average) judge that medicine $B$ has no effect in the first experimental condition, but $B$ does have an effect in the second condition. By contrast, the linear model makes the opposite (wrong) prediction. In summary, model selection comparing two noisy-logical models gives a good prediction of participant performance.

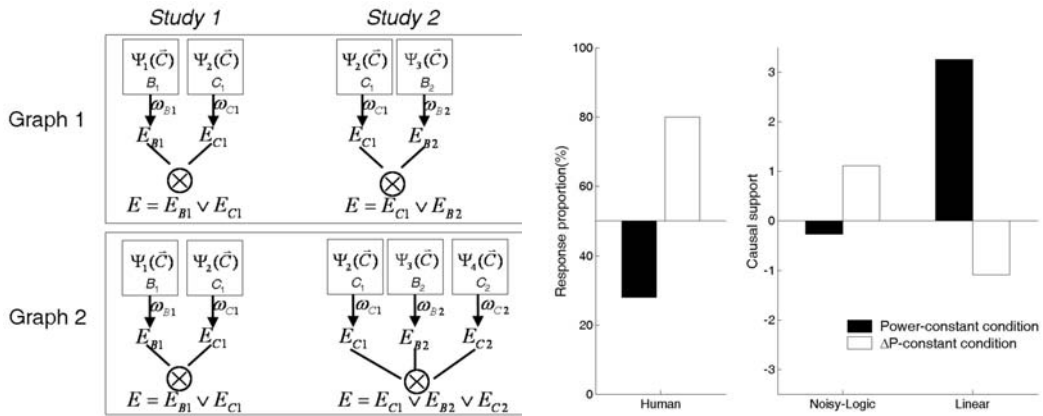

Figure 3: Causal model and results for Experiment I. Left panel: two alternative causal models for the two studies. Right panel: the experimental results (proportion of patients who think medicine $B$ causes headaches)) for the Power-constant and $\Delta P$-constant conditions [6]. Far right, the causal support for the noisy-logic and linear models.

## 5.3 Experiment II: Causal Interaction

Liljeholm and Cheng [6] also investigated causal interactions. The experimental design was identical to that used in Experiment 1, except that participants were presented with three studies in which only one medicine ($A$) was tested. Participants were asked to judge whether medicine $A$ interacts with background causes that vary across the three studies. We define the background causes as $B_1, B_2, B_3$ for the three studies, and $C_1$ for medicine $A$. This experiment was also run under two different conditions, see table (2). The first *power-constant condition* [6] was consistent with a noisy-logical model, but the second *power-varying condition* [6] was not.

Table 2: Experimental conditions (1) and (2) for Experiment 2

| | | |
|---|---|---|
| (1) | $P(E=1\|B_1=1, C_1=0), P(E=1\|B_1=1, C_1=1)$ | 16/24, 22/24 |
| | $P(E=1\|B_2=1, C_1=0), P(E=1\|B_2=1, C_1=1)$ | 8/24, 20/24 |
| | $P(E=1\|B_3=1, C_1=0), P(E=1\|B_3=1, C_1=1)$ | 0/24, 18/24 |
| (2) | $P(E=1\|B_1=1, C_1=0), P(E=1\|B_1=1, C_1=1)$ | 0/24, 6/24 |
| | $P(E=1\|B_2=1, C_1=0), P(E=1\|B_2=1, C_1=1)$ | 0/24, 12/24 |
| | $P(E=1\|B_3=1, C_1=0), P(E=1\|B_3=1, C_1=1)$ | 0/24, 18/24 |

The experimental design caused participants to choose between two causal models shown on the left panel of figure (4). The probability of generating the data is given by $P(D|\vec{\omega}, Graph) = P(D_1|\vec{\omega}_1, Graph)P(D_2|\vec{\omega}_2, Graph)P(D_3|\vec{\omega}_3, Graph)$. For $Graph1$, the $P(D_i|.)$ are noisy-or distributions $P(E|B_1, C_1, \omega_{B1}, \omega_{C1}), P(E|B_2, C_1, \omega_{B2}, \omega_{C1}), P(E|B_3, C_1, \omega_{B3}, \omega_{C1})$. For $Graph2$, the $P(D_i|.)$ are $P(E|B_1, C_1, \omega_{B1}, \omega_{C1}), P(E|B_2, C_1, B_2C_1, \omega_{B2}, \omega_{C1}, \omega_{B2C1})$ and $P(E|B_3, C_1, B_3C_1, \omega_{B3}, \omega_{C1}, \omega_{B3C1})$.

All the distributions are noisy-or on the unary causal features (e.g. $B, C_1$), but the nature of the conjunctive cause $B \wedge C_1$ is unknown (i.e. not specified by the experimental design). Hence our theory considers the possibilities that it is a noisy-or (e.g. can produce headaches) or noisy-and-not (e.g. can prevent headaches), see graph 2 of Figure (4).

## 5.4 Results of Experiment II

Figure (4) shows human and model performance for the two experimental conditions. Our noisy-logical model is in agreement with human performance – i.e. there is no interaction between causes in the power-constant condition, but there is interaction in the power-varying condition. By contrast, the linear model predicts interaction in both conditions and hence fails to model human performance.

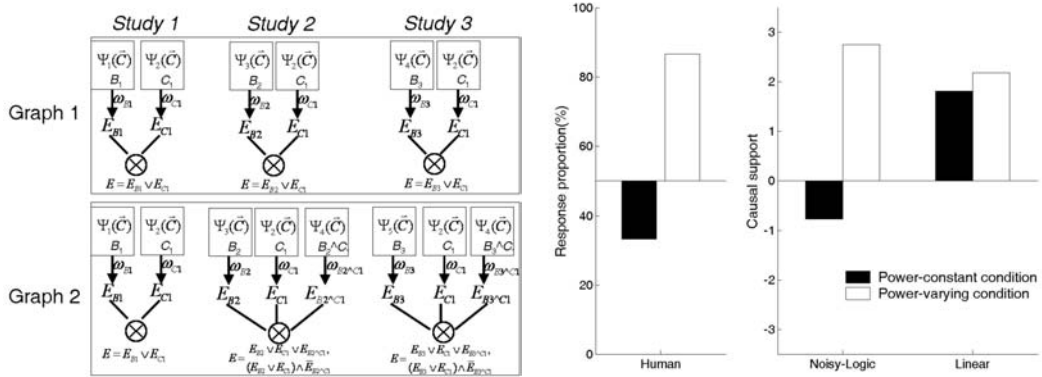

Figure 4: Causal model and results for Experiment II. Left panel: two alternative causal models (one involving conjunctions) for the three studies . Right panel: the proportion of participants who think that there is an interaction (conjunction) between medicine $A$ and the background for the power-constant and power-varying conditions [6]. Far right, the causal support for the noisy-logical and linear models.

# 6 Summary

The noisy-logical distribution gives a new way to represent conditional probability distributions defined over binary variables. The complexity of the distribution can be adjusted by restricting the set of causal factors. If all the causal factors are allowed, then the distribution can represent any conditional distribution. But by restricting the set of causal factors we can obtain standard distributions such as the noisy-or and noisy-and-not.

We illustrated the noisy-logical distribution by modeling experimental findings on causal reasoning. Our results showed that this distribution fitted the experimental data and, in particular, accounted for the major trends (unlike the linear model). This is consistent with the success of noisy-or and noisy-and-not models for accounting for experiments involving two causes [1], [2],[3]. This suggests that humans may make use of noisy-logical representations for causal reasoning.

One attraction of the noisy-logical representation is that it helps clarify the relationship between logic and probabilities. Standard logical relationships between causes and effects arise in the limit as the $\omega_i$ take values $0$ or $1$. We can, for example, bias the data towards a logical form by using a prior on the $\vec{\omega}$. This may be useful, for example, when modeling human cognition – evidence suggests that humans first learn logical relationships and, only later, move to probabilities.

In summary, the noisy-logical distribution is a novel way to represent conditional probability distributions defined on binary variables. We hope this class of distributions will be useful for modeling cognitive phenomena and for applications to artificial intelligence.

## Acknowledgements

We thank Mimi Liljeholm, Patricia Cheng, Adnan Darwiche, Keith Holyoak, Iasonas Kokkinos, and YingNian Wu for helpful discussions. Mimi and Patricia kindly gave us access to their experimental data. We acknowledge funding support from the W.M. Keck foundation and from NSF 0413214.

## References

[1] P. W. Cheng. From covariation to causation: A causal power theory. Psychological Review, 104, 367405. 1997.

[2] L.R. Novick and P.W. Cheng. Assessing interactive causal influence. Psychological Review, 111, 455-485. 2004.

[3] T. L. Griffiths, and J. B. Tenenbaum. Structure and strength in causal induction. Cognitive Psychology, 51, 334-384, 2005.

[4] J. Pearl, Probabilistic Reasoning in Intelligent Systems. Morgan-Kauffman, 1988.

[5] C.N. Glymour. The Mind's Arrow: Bayes Nets and Graphical Causal Models in Psychology. MIT Press. 2001.

[6] M. Liljeholm and P. W. Cheng. When is a Cause the "Same"? Coherent Generalization across Contexts. Psychological Science, in press. 2007.

[7] M. J. Buehner, P. W. Cheng, and D. Clifford. From covariation to causation: A test of the assumption of causal power. Journal of Experimental Psychology: Learning, Memory, and Cognition, 29, 1119-1140, 2003.

[8] R. A. Rescorla, and A. R. Wagner. A theory of Pavlovian conditioning: Variations in the effectiveness of reinforcement and nonreinforcement. In A. H. Black and W. F. Prokasy (Eds.), Classical conditioning II: Current theory and research (pp. 64-99). New York: Appleton-Century Crofts. 1972.

